# Spike and Slab Variational Inference for Multi-Task and Multiple Kernel Learning

**Michalis K. Titsias**
University of Manchester
mtitsias@gmail.com

**Miguel Lázaro-Gredilla**
Univ. de Cantabria & Univ. Carlos III de Madrid
miguel@tsc.uc3m.es

## Abstract

We introduce a variational Bayesian inference algorithm which can be widely applied to sparse linear models. The algorithm is based on the spike and slab prior which, from a Bayesian perspective, is the golden standard for sparse inference. We apply the method to a general multi-task and multiple kernel learning model in which a common set of Gaussian process functions is linearly combined with task-specific sparse weights, thus inducing relation between tasks. This model unifies several sparse linear models, such as generalized linear models, sparse factor analysis and matrix factorization with missing values, so that the variational algorithm can be applied to all these cases. We demonstrate our approach in multi-output Gaussian process regression, multi-class classification, image processing applications and collaborative filtering.

## 1 Introduction

Sparse inference has found numerous applications in statistics and machine learning [1, 2, 3]. It is a generic idea that can be combined with popular models, such as linear regression, factor analysis and more recently multi-task and multiple kernel learning models. In the regularization theory literature sparse inference is tackled via $\ell_1$ regularization [2], which requires expensive cross-validation for model selection. From a Bayesian perspective, the spike and slab prior [1, 4, 5], also called two-groups prior [6], is the golden standard for sparse linear models. However, the discrete nature of the prior makes Bayesian inference a very challenging problem. Specifically, for $M$ linear weights, inference under a spike and slab prior distribution on those weights requires a combinatorial search over $2^M$ possible models. The problems found when working with the spike and slab prior led several researchers to consider soft-sparse or shrinkage priors such as the Laplace and other related scale mixtures of normals [3, 7, 8, 9, 10]. However, such priors are not ideal since they assign zero probability mass to events associated with weights having zero value.

In this paper, we introduce a simple and efficient variational inference algorithm based on the spike and slab prior which can be widely applied to sparse linear models. The novel characteristic of this algorithm is that the variational distribution over sparse weights has a factorial nature, i.e., it can be written as a mixture of $2^M$ components where $M$ is the number of weights. Unlike the standard mean field approximation which uses a unimodal variational distribution, our variational algorithm can more precisely match the combinational nature of the posterior distribution over the weights. We will show that the proposed variational approach is more accurate and robust to unfavorable initializations than the standard mean field variational approximation.

We apply the variational method to a general multi-task and multiple kernel learning model that expresses the correlation between tasks by letting them share a common set of Gaussian process latent functions. Each task is modeled by linearly combining these latent functions with task-specific weights which are given a spike and slab prior distribution. This model is a spike and slab Bayesian reformulation of previous Gaussian process-based single-task multiple kernel learning

methods [11, 12, 13] and multi-task Gaussian processes (GPs) [14, 15, 16, 17]. Further, this model unifies several sparse linear models, such as generalized linear models, factor analysis, probabilistic PCA and matrix factorization with missing values. In the experiments, we apply the variational inference algorithms to all the above models and present results in multi-output regression, multi-class classification, image denoising, image inpainting and collaborative filtering.

## 2 Spike and slab multi-task and multiple kernel learning

Section 2.1 discusses the spike and slab multi-task and multiple kernel learning (MTMKL) model that linearly combines Gaussian process latent functions. Spike and slab factor analysis and probabilistic PCA is discussed in Section 2.2, while missing values are dealt with in Section 2.3.

### 2.1 The model

Let $\mathcal{D} = \{\mathbf{X}, \mathbf{Y}\}$, with $\mathbf{X} \in \mathbb{R}^{N \times D}$ and $\mathbf{Y} \in \mathbb{R}^{N \times Q}$, be a dataset such that the $n$-th row of $\mathbf{X}$ is an input vector $\mathbf{x}_n$ and the $n$-th row of $\mathbf{Y}$ is the set of $Q$ corresponding tasks or outputs. We use $\mathbf{y}_q$ to refer to the $q$-th column of $\mathbf{Y}$ and $y_{nq}$ to the $(n, q)$ entry. Outputs $\mathbf{Y}$ are then assumed to be generated according to the following hierarchical Bayesian model:

$$y_{nq} \sim \mathcal{N}(y_{nq}|f_q(\mathbf{x}_n), \sigma_q^2), \qquad \forall_{n,q} \qquad (1a)$$

$$f_q(\mathbf{x}) = \sum_{m=1}^{M} w_{qm}\phi_m(\mathbf{x}) = \mathbf{w}_q^\top \phi(\mathbf{x}), \qquad \forall_q \qquad (1b)$$

$$w_{qm} \sim \pi\mathcal{N}(w_{qm}|0, \sigma_w^2) + (1-\pi)\delta_0(w_{qm}), \qquad \forall_{q,m} \qquad (1c)$$

$$\phi_m(\mathbf{x}) \sim \mathcal{GP}(\mu_m(\mathbf{x}), k_m(\mathbf{x}_i, \mathbf{x}_j)), \qquad \forall_m. \qquad (1d)$$

Here, each $\mu_m(\mathbf{x})$ is a mean function, $k_m(\mathbf{x}_i, \mathbf{x}_j)$ a covariance function, $\mathbf{w}_q = [w_{q1}, \dots, w_{qM}]^\top$, $\phi(\mathbf{x}) = [\phi_1(\mathbf{x}), \dots, \phi_M(\mathbf{x})]^\top$ and $\delta_0(w_{qm})$ denotes the Dirac delta function centered at zero. Since each of the $Q$ tasks is a linear combination of the same set of latent functions $\{\phi_m(\mathbf{x})\}_{m=1}^{M}$ (where typically $M < Q$), correlation is induced in the outputs. Sharing a common set of features means that "knowledge transfer" between tasks can occur and latent functions are inferred more accurately, since data belonging to all tasks are used.

Several linear models can be expressed as special cases of the above. For instance, a generalized linear model is obtained when the GPs are Dirac delta measures (with zero covariance functions) that deterministically assign each $\phi_m(\mathbf{x})$ to its mean function $\mu_m(\mathbf{x})$. However, the model in (1) has a number of additional features not present in standard linear models. Firstly, the basis functions are no longer deterministic, but they are instead drawn from different GPs, so an extra layer of flexibility is added to the model. Thus, a posterior distribution over the basis functions of the generalized linear model can be inferred from data. Secondly, a truly sparse prior, the spike and slab prior (1c), is placed over the weights of the model. Specifically, with probability $1-\pi$, each $w_{qm}$ is zero, and with probability $\pi$, it is drawn from a Gaussian. This contrasts with previous approaches [3, 7, 8, 9, 13] in which soft-sparse priors that assign zero probability mass to the weights being exactly zero were used. Hyperparameters $\pi$ and $\sigma_w^2$ are learnable in order to determine the amount of sparsity and the discrepancy of nonzero weights, respectively. Thirdly, the number of basis functions $M$ can be inferred from data, since the sparse prior on the weights allows basis functions to be "switched off" as necessary by setting the corresponding weights to zero.

Further, the model in (1) can be considered as a spike and slab Bayesian reformulation of multi-task [14, 15] and multiple kernel learning previous methods [11, 12] that learn the weights using maximum likelihood. By assuming the weights $\mathbf{w}_q$ are given, each output function $y_q(\mathbf{x})$ is a GP with covariance function

$$\text{Cov}[(y_q(\mathbf{x}_i), y_q(\mathbf{x}_j)] = \sum_{m=1}^{M} w_{qm}^2 k_m(\mathbf{x}_i, \mathbf{x}_j),$$

which clearly consists of a conic combination of kernel functions. Therefore, the proposed model can be reinterpreted as multiple kernel learning in which the weights of each kernel are assigned spike and slab priors in a full Bayesian formulation.

## 2.2 Sparse factor and principal component analysis

An interesting case arises when $\mu_m(\mathbf{x}) = 0$ and $k_m(\mathbf{x}_i, \mathbf{x}_j) = \delta_{ij} \; \forall m$, where $\delta_{ij}$ is the Kronecker delta. This says that each latent function is drawn from a white process so that it consists of independent values each following the standard normal distribution. We first define matrices $\boldsymbol{\Phi} \in \mathbb{R}^{N \times M}$ and $\mathbf{W} \in \mathbb{R}^{Q \times M}$, whose elements are, respectively, $\phi_{nm} = \phi_m(\mathbf{x}_n)$ and $w_{qm}$. Then, the model in (1) reduces to

$$\mathbf{Y} = \boldsymbol{\Phi}\mathbf{W}^\top + \boldsymbol{\xi}, \tag{2a}$$

$$w_{qm} \sim \pi \mathcal{N}(w_{qm}|0, \sigma_w^2) + (1-\pi)\delta_0(w_{qm}), \qquad \forall_{q,m} \tag{2b}$$

$$\phi_{nm} \sim \mathcal{N}(\phi_{nm}|0, 1), \qquad \forall_{n,m} \tag{2c}$$

$$\xi_{nq} \sim \mathcal{N}(\xi_{nq}|0, \sigma_q^2), \qquad \forall_{n,q}, \tag{2d}$$

where $\boldsymbol{\xi}$ is an $N \times Q$ noise matrix with entries $\xi_{nq}$. The resulting model thus corresponds to sparse factor analysis or sparse probabilistic PCA (when the noise is homoscedastic, i.e., $\sigma_q^2$ is constant for all $q$). Observe that the sparse spike and slab prior is placed on the factor loadings $\mathbf{W}$.

## 2.3 Missing values

The method can easily handle missing values and thus be applied to problems involving matrix completion and collaborative filtering. More precisely, in the presence of missing values we have a binary matrix $\mathbf{Z} \in \mathbb{R}^{N \times Q}$ that indicates the observed elements in $\mathbf{Y}$. Using $\mathbf{Z}$ the likelihood in (1a) is modified according to $y_{nq} \sim \mathcal{N}(y_{nq}|f_q(\mathbf{x}_n), \sigma_q^2)$, $\forall_{n,q}$ s.t. $[\mathbf{Z}]_{nq} = 1$. In the experiments we consider missing values in applications such as image inpainting and collaborative filtering.

# 3 Efficient variational inference

The presence of the Dirac delta mass function makes the application of variational approximate inference algorithms in spike and slab Bayesian models troublesome. However, there exists a simple reparameterization of the spike and slab prior that is more amenable to approximate inference methods. Specifically, assume a Gaussian random variable $\widetilde{w}_{qm} \sim \mathcal{N}(\widetilde{w}_{qm}|0, \sigma_w^2)$ and a Bernoulli random variable $s_{qm} \sim \pi^{s_{qm}}(1-\pi)^{1-s_{qm}}$. The product $s_{qm}\widetilde{w}_{qm}$ forms a new random variable that follows the probability distribution in eq. (1c). This allows to reparameterize $w_{qm}$ according to $w_{qm} = s_{qm}\widetilde{w}_{qm}$ and assign the above prior distributions on $s_{qm}$ and $\widetilde{w}_{qm}$. Thus, the reparameterized spike and slab prior takes the form:

$$p(\widetilde{w}_{qm}, s_{qm}) = \mathcal{N}(w_{qm}|0, \sigma_w^2)\pi^{s_{qm}}(1-\pi)^{1-s_{qm}}, \qquad \forall_{q,m}. \tag{3}$$

Notice that the presence of $w_{qm}$ in the likelihood function in (1a) is now replaced by the product $s_{qm}\widetilde{w}_{qm}$. After the above reparameterization, a standard mean field variational method that uses the factorized variational distribution over $\widetilde{\mathbf{W}} = \{\widetilde{\mathbf{w}}_q\}_{q=1}^Q$ and $\mathbf{S} = \{\mathbf{s}_q\}_{q=1}^Q$ takes the form $q(\widetilde{\mathbf{W}}, \mathbf{S}) = \prod_{q=1}^Q q(\widetilde{\mathbf{w}}_q, \mathbf{s}_q)$, where

$$q(\widetilde{\mathbf{w}}_q, \mathbf{s}_q) = q(\widetilde{\mathbf{w}}_q)q(\mathbf{s}_q) = \mathcal{N}(\widetilde{\mathbf{w}}_q|\boldsymbol{\mu}_{w_q}, \boldsymbol{\Sigma}_{w_q}) \prod_{m=1}^M \gamma_{qm}^{s_{qm}}(1-\gamma_{qm})^{1-s_{qm}} \tag{4}$$

and where $(\boldsymbol{\mu}_{w_q}, \Sigma_{w_q}, \boldsymbol{\gamma}_q)$ are variational parameters. Such an approach has extensively used in [18] and also considered in [19]. However, the above variational distribution leads to a very inefficient approximation. This is because (4) is a unimodal distribution, and therefore has limited capacity when approximating the factorial true posterior distribution which can have exponentially many modes. To analyze the nature of the true posterior distribution, we consider the following two properties derived by assuming for simplicity a single output ($Q = 1$) so index $q$ is dropped.

**Property 1:** The true marginal posterior $p(\widetilde{\mathbf{w}}|\mathbf{Y})$ can be written as a mixture distribution having $2^M$ components. This is an obvious fact since $p(\widetilde{\mathbf{w}}|\mathbf{Y}) = \sum_{\mathbf{s}} p(\widetilde{\mathbf{w}}|\mathbf{s}, \mathbf{Y})p(\mathbf{s}|\mathbf{Y})$, where the summation involves all $2^M$ possible values of the binary vector $\mathbf{s}$.

The second property characterizes the nature of each conditional $p(\widetilde{\mathbf{w}}|\mathbf{s}, \mathbf{Y})$ in the above sum.

**Property 2:** Assume the conditional distribution $p(\widetilde{\mathbf{w}}|\mathbf{s}, \mathbf{Y})$. We can write $\mathbf{s} = \mathbf{s}_1 \cup \mathbf{s}_0$, where $\mathbf{s}_1$ denotes the elements in $\mathbf{s}$ with value one and $\mathbf{s}_0$ the elements with value zero. Using the

correspondence between $\mathbf{s}$ and $\widetilde{\mathbf{w}}$, we have $\widetilde{\mathbf{w}} = \widetilde{\mathbf{w}}_1 \cup \widetilde{\mathbf{w}}_0$. Then, $p(\widetilde{\mathbf{w}}|\mathbf{s}, \mathbf{Y})$ factorizes as $p(\widetilde{\mathbf{w}}|\mathbf{s}, \mathbf{Y}) = p(\widetilde{\mathbf{w}}_1|\mathbf{Y})\mathcal{N}(\widetilde{\mathbf{w}}_0|0, \sigma_w^2 I_{|\widetilde{\mathbf{w}}_0|})$, which says that the posterior over $\widetilde{\mathbf{w}}_0$ given $\mathbf{s}_0 = \mathbf{0}$ is equal to the prior over $\widetilde{\mathbf{w}}_0$. This property is obvious because $\widetilde{\mathbf{w}}_0$ and $\mathbf{s}_0$ appear in the likelihood as an elementwise product $\widetilde{\mathbf{w}}_0 \circ \mathbf{s}_0$, thus when $\mathbf{s}_0 = \mathbf{0}$, $\widetilde{\mathbf{w}}_0$ becomes disconnected from the data.

The standard variational distribution in (4) ignores these properties and approximates the marginal $p(\widetilde{\mathbf{w}}|\mathbf{Y})$, which is a mixture with $2^M$ components, with a single Gaussian distribution. Next we present an alternative variational approximation that takes into account the above properties.

## 3.1 The proposed variational method

In the reparameterized spike and slab prior, each pair of variables $\{\widetilde{w}_{qm}, s_{qm}\}$ is strongly correlated since their product is the underlying variable that interacts with the data. Thus, a sensible approximation must treat each pair $\{\widetilde{w}_{qm}, s_{qm}\}$ as a unit so that $\{\widetilde{w}_{qm}, s_{qm}\}$ are placed in the same factor of the variational distribution. The simplest factorization that achieves this is:

$$q(\widetilde{\mathbf{w}}_q, \mathbf{s}_q) = \prod_{m=1}^{M} q(\widetilde{w}_{qm}, s_{qm}). \tag{5}$$

This variational distribution yields a marginal $q(\widetilde{\mathbf{w}}_q)$ which has $2^M$ components. This can be seen by writing $q(\widetilde{\mathbf{w}}_q) = \prod_{m=1}^{M} [q(\widetilde{w}_{qm}, s_{qm} = 1) + q(\widetilde{w}_{qm}, s_{qm} = 0)]$ and then by multiplying the terms a mixture of $2^M$ components is obtained. Therefore, **Property 1** is satisfied by (5). In turns out that **Property 2** is also satisfied. This can be shown by taking the stationary condition for the factor $q(\widetilde{w}_{qm}, s_{qm})$ when maximizing the variational lower bound (on the true marginal likelihood):

$$\left\langle \log \frac{p(\mathbf{Y}, \widetilde{w}_{qm}, s_{qm}, \Theta)p(\Theta)\mathcal{N}(\widetilde{w}_{qm}|0, \sigma_w^2)\pi^{s_{qm}}(1-\pi)^{1-s_{qm}}}{q(\widetilde{w}_{qm}, s_{qm})q(\Theta)} \right\rangle_{q(\widetilde{w}_{qm}, s_{qm})q(\Theta)}, \tag{6}$$

where $\Theta$ are the remaining random variables in the model (i.e., excluding $\{\widetilde{w}_{qm}, s_{qm}\}$) and $q(\Theta)$ their variational distribution. The stationary condition for $q(\widetilde{w}_{qm}, s_{qm})$ is

$$q(\widetilde{w}_{qm}, s_{qm}) = \frac{1}{\mathcal{Z}} e^{\langle \log p(\mathbf{Y}, \widetilde{w}_{qm}, s_{qm}, \Theta) \rangle_{q(\Theta)}} \mathcal{N}(\widetilde{w}_{qm}|0, \sigma_w^2)\pi^{s_{qm}}(1-\pi)^{1-s_{qm}}, \tag{7}$$

where $\mathcal{Z}$ is a normalizing constant that does not depend on $\{\widetilde{w}_{qm}, s_{qm}\}$. Therefore, we have $q(\widetilde{w}_{qm}|s_{qm} = 0) \propto q(\widetilde{w}_{qm}, s_{qm} = 0) = \frac{\mathcal{C}}{\mathcal{Z}}\mathcal{N}(\widetilde{w}_{qm}|0, \sigma_w^2)(1-\pi)$, where $\mathcal{C} = e^{\langle \log p(\mathbf{Y}, \widetilde{w}_{qm}, s_{qm}=0, \Theta) \rangle_{q(\Theta)}}$ is a constant that does not depend on $\widetilde{w}_{qm}$. From the last expression we obtain $q(\widetilde{w}_{qm}|s_{qm} = 0) = \mathcal{N}(\widetilde{w}_{qm}|0, \sigma_w^2)$ which implies that **Property 2** is satisfied.

The above remarks regarding variational distribution (5) are general and can hold for many spike and slab probability models as long as the weights $\widetilde{\mathbf{w}}$ and binary variables $\mathbf{s}$ interact inside the likelihood function according to $\widetilde{\mathbf{w}} \circ \mathbf{s}$.

## 3.2 Application to the multi-task and multiple kernel learning model

Here, we briefly discuss the variational method applied to the multi-task and multiple kernel model described in Section 2.1 and refer to supplementary material for variational EM update equations.

The explicit form of the joint probability density function on the training data of model (1) is

$$p(\mathbf{Y}, \widetilde{\mathbf{W}}, \mathbf{S}, \mathbf{\Phi}) = \mathcal{N}(\mathbf{Y}|\mathbf{\Phi}(\widetilde{\mathbf{W}} \circ \mathbf{S})^\top, \mathbf{\Sigma}) \prod_{q,m} \left[ \mathcal{N}(\widetilde{w}_{qm}|0, \sigma_w^2)\pi^{s_{qm}}(1-\pi)^{s_{qm}} \right] \prod_{m=1}^{M} \mathcal{N}(\phi_m|\boldsymbol{\mu}_m, \mathbf{K}_m),$$

where $\{\widetilde{\mathbf{W}}, \mathbf{S}, \mathbf{\Phi}\}$ is the whole set of random variables that need to be marginalized out to compute the marginal likelihood. The marginal likelihood is analytically intractable, so we lower bound it using the following variational distribution

$$q(\widetilde{\mathbf{W}}, \mathbf{S}, \mathbf{\Phi}) = \prod_{q=1}^{Q} \prod_{m=1}^{M} q(\widetilde{w}_{qm}, s_{qm}) \prod_{m=1}^{M} q(\phi_m). \tag{8}$$

The stationary conditions of the lower bound result in analytical updates for all factors above. More precisely, $q(\boldsymbol{\phi}_m)$ is an $N$-dimensional Gaussian distribution and each factor $q(\widetilde{w}_{qm}, s_{qm})$ leads to a marginal $q(\widetilde{w}_{qm})$ which is a mixture of two Gaussians where one component is $q(\widetilde{w}_{qm}|s_{qm} = 0) = \mathcal{N}(\widetilde{w}_{qm}|0, \sigma_w^2)$, as shown in the previous section. The optimization proceeds using an EM algorithm that at the E-step updates the factors in (8) and at the M-step updates hyperparameters $\{\{\sigma_q\}_{q=1}^Q, \sigma_w^2, \pi, \{\boldsymbol{\theta}_m\}_{m=1}^M\}$ where $\boldsymbol{\theta}_m$ parameterize kernel matrix $\mathbf{K}_m$. There is, however, one surprise in these updates. The GP hyperparameters $\boldsymbol{\theta}_m$ are strongly dependent on the factor $q(\boldsymbol{\phi}_m)$ of the corresponding GP latent vector, so updating $\boldsymbol{\theta}_m$ by keeping fixed the factor $q(\boldsymbol{\phi}_m)$ exhibits slow convergence. This problem is efficiently resolved by applying a Marginalized Variational step [20] which jointly updates the pair $(q(\boldsymbol{\phi}_m), \boldsymbol{\theta}_m)$. This more advanced update together with all remaining updates of the EM algorithm are discussed in detail in the supplementary material.

## 4   Assessing the accuracy of the approximation

In this section we compare the proposed variational inference method, in the following called paired mean field (PMF), against the standard mean field (MF) approximation. For simplicity, we consider a single-output linear regression problem where the data are generated according to: $\mathbf{y} = (\widetilde{\mathbf{w}} \circ \mathbf{s})^T \mathbf{x} + \boldsymbol{\xi}$. Moreover, to remove the effect of hyperparameter learning from the comparison, $(\sigma^2, \pi, \sigma_w^2)$ are fixed to known values. The objective of the comparison is to measure the accuracy when approximating the true posterior mean value for the parameter vector $\mathbf{w}^{\text{tr}} = \mathbb{E}[\widetilde{\mathbf{w}} \circ \mathbf{s}]$ where the expectation is under the true posterior distribution. $\mathbf{w}^{\text{tr}}$ is obtained by running a very long run of Gibbs sampling. PMF and MF provide alternative approximations $\mathbf{w}^{\text{PMF}}$ and $\mathbf{w}^{\text{MF}}$, and absolute errors between these approximations and $\mathbf{w}^{\text{tr}}$ are used to measure accuracy. Since initialization is crucial for variational non-convex algorithms, the accuracy of PMF and MF is averaged over many random initializations of their respective variational distributions.

|     | *soft*-error | *soft*-bound | *extreme*-error | *extreme*-bound |
| --- | --- | --- | --- | --- |
| MF | 0.917 [0.002,1.930] | -628.9 [-554.6,-793.5] | 1.880 [0.965, 2.561] | -895.0 [-618.9,-1483.3] |
| PMF | 0.208 [0.002,0.454] | -560.7 [-557.8, -564.1] | 0.204 [0.002, 0.454] | -560.6 [-557.8, -564.0] |

Table 1: Comparison of MF and PMF in Boston-housing data in terms of approximating the ground-truth. Average errors ($\sum_{m=1}^{13} |w_m^{\text{tr}} - w_m^{\text{appr}}|$) together with 95% confidence intervals (given by percentiles) are shown for *soft* and *extreme* initializations. Average values for the variational lower bound are also shown.

For the purpose of the comparison we also derived an efficient paired Gibbs sampler that follows exactly the same principle as PMF. This Gibbs sampler iteratively samples the pair $(\widetilde{w}_m, s_m)$ from the conditional $p(\widetilde{w}_m, s_m|\widetilde{\mathbf{w}}_{\backslash m}, \mathbf{s}_{\backslash m}, \mathbf{y})$ and has been observed to mix much faster than the standard Gibbs sampler that samples $\widetilde{\mathbf{w}}$ and $\mathbf{s}$ separately. More details about the paired Gibbs sampler are given in the supplementary material.

We considered the Boston-housing dataset which consists of 456 training examples and 13 inputs. Hyperparameters were fixed to values ($\sigma^2 = 0.1 \times \text{var}(\mathbf{y}), \pi = 0.25, \sigma_w^2 = 1$) where $\text{var}(\mathbf{y})$ denotes the variance of the data. We performed two types of experiments each repeated 300 times. Each repetition of the first type uses a *soft* random initialization of each $q(s_m = 1) = \gamma_m$ from the range $(0, 1)$. The second type uses an *extreme* random initialization so that each $\gamma_m$ is initialized to either 0 or 1. For each run PMF and MF are initialized to the same variational parameters.

Table 1 reports average absolute errors and also average values of the variational lower bounds. Clearly, PMF is more accurate than MF, achieves significantly higher values for the lower bound and exhibits smaller variance under different initializations. Further, for the more difficult case of *extreme* initializations the performance of MF becomes worse, while the performance of PMF remains unchanged. This shows that optimization in PMF, although non-convex, is very robust to unfavorable initializations. Similar experiments in other datasets have confirmed the above remarks.

## 5   Experiments

**Toy multi-output regression dataset.** To illustrate the capabilities of the proposed model, we first apply it to a toy multi-output dataset with missing observations. Toy data is generated as follows:

Ten random latent functions are generated by sampling i.i.d. from zero-mean GPs with the following non-stationary covariance function

$$k(x_i, x_j) = \exp\left(\frac{-x_i^2 - x_j^2}{20}\right)(4\cos(0.5(x_i - x_j)) + \cos(2(x_i - x_j))),$$

at 201 evenly spaced points in the interval $x \in [-10, 10]$. Ten tasks are then generated by adding Gaussian noise with standard deviation 0.2 to those random latent functions, and two additional tasks consist only of Gaussian noise with standard deviations 0.1 and 0.4. Finally, for each of the 12 tasks, we artificially simulate missing data by removing 41 contiguous observations, as shown in Figure 1. Missing data are not available to any learning algorithm, and will be used to test performance only. Note that the above covariance function is rank-4, so ten out of the twelve tasks will be related, though we do not know how, or which ones.

All tasks are then learned using both independent GPs with squared exponential (SE) covariance function $k_{\text{SE}}(x_i, x_j) = \exp(-(x_i - x_j)^2/(2\ell))$ and the proposed MTMKL with $M = 7$ latent functions, each of them also using the SE prior. Hyperparameter $\ell$, as well as noise levels are learned independently for each latent function. Figure 1 shows the inferred posterior means.

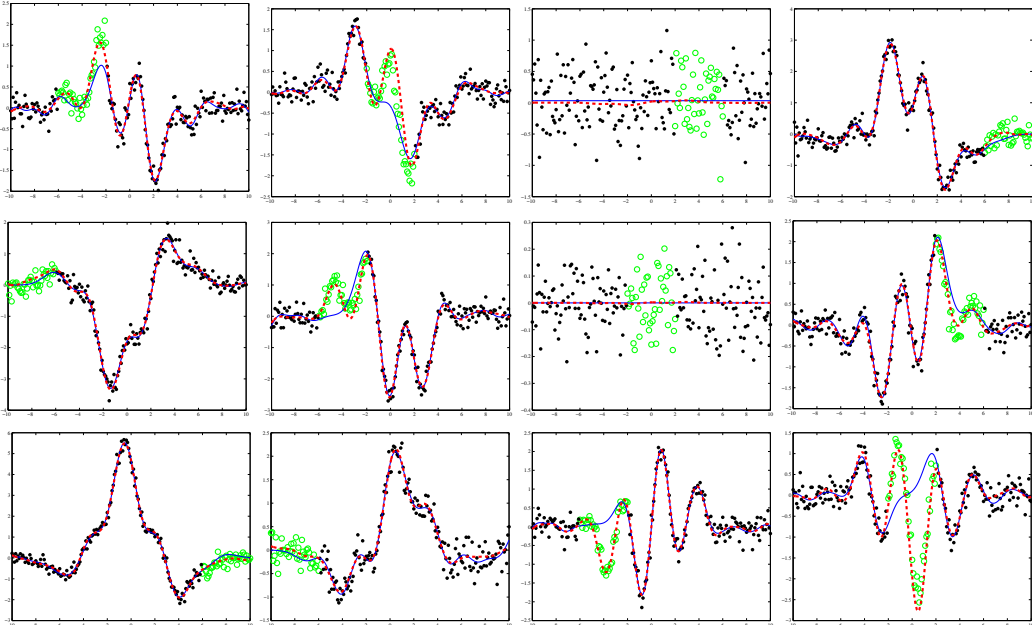

Figure 1: Twelve related tasks and predictions according to independent GPs (blue, continuous line) and MTMKL (red, dashed line). Missing data for each task is represented using green circles.

The mean square error (MSE) between predictions and missing observations for each task are displayed in Table 2. MTMKL is able to infer how tasks are related and then exploit that information to make much better predictions. After learning, only 4 out of the 7 available latent functions remain active, while the other ones are pruned by setting the corresponding weights to zero. This is in correspondence with the generating covariance function, which only had 4 eigenfunctions, showing how model order selection is automatic.

| Method \ Task # | 1 | 2 | 3 | 4 | 5 | 6 | 7 | 8 | 9 | 10 | 11 | 12 |
|---|---|---|---|---|---|---|---|---|---|---|---|---|
| Independent GPs | 6.51 | 11.70 | 7.52 | 2.49 | 1.53 | 18.25 | 0.41 | 7.43 | 2.73 | 1.81 | 19.93 | 93.80 |
| MTMKL | 1.97 | 4.57 | 7.71 | 1.94 | 1.98 | 2.09 | 0.41 | 1.96 | 1.90 | 1.57 | 1.20 | 2.83 |

Table 2: MSE performance of independent GPs vs. MTMKL on the missing observations for each task.

Inferred noise standard deviations for the noise-only tasks are 0.10 and 0.45, and the average for the remaining tasks is 0.22, which agrees well with the stated actual values.

**The flowers dataset.** Though the proposed model has been designed as a tool for regression, it can also be used approximately to solve classification problems by using output values to identify class membership. In this section we will apply it to the challenging flower identification problem posed in [21]. There are 2040 instances of flowers for training and 6149 for testing, mainly acquired from the web, with varying scales, resolutions, etc., which are labeled into 102 categories. In [21], four relevant features are identified: Color, histogram of gradient orientations and the scale invariant feature transform, sampled on both the foreground region and its boundary. More information is available at http://www.robots.ox.ac.uk/~vgg/data/flowers/.

For this type of dataset, state of the art performance has been achieved using a weighted linear combination of kernels (one per feature) in a support vector machine (SVM) classifier. A different set of weights is learned for each class. In [22] it is shown that these weights can be learned by solving a convex optimization problem. I.e., the standard approach to tackle the flower classification problem would correspond to solving 102 independent binary classification problems, each using a linear combination of 4 kernels. We take a different approach: Since all the 102 binary classification tasks are related, we learn all of them at once as a multi-task multiple-kernel problem, hoping that knowledge transfer between them will enhance performance.

For each training instance, we set the corresponding output to +1 for the desired task, whereas the output for the remaining tasks is set to -1. Then we consider both using 10 and 13 latent functions per feature (i.e., $M = 40$ and $M = 52$). We measure performance in terms of the recognition rate (RR), which is the average of break-even points (where precision equals recall) for each class; average area under the curve (AUC); and the multi-class accuracy (MA) which is the rate of correctly classified instances. As baseline, recall that a random classifier would yield a RR and AUC of 0.5 and a MA of $1/102 = 0.0098$. Results are reported in Table 3.

| Method | Latent function # | AUC on test set | RR on test set | MA on test set |
|---|---|---|---|---|
| MTMKL | $M = 40$ | 0.944 | 0.889 | 0.329 |
| MTMKL | $M = 52$ | 0.952 | 0.893 | 0.400 |
| MKL from [21] | $M = 408$ | - | 0.728 | - |
| MKL from [13] | $M = 408$ | 0.957 | - | - |

Table 3: Performance of the different multiple kernel learning algorithms on the flowers dataset.

MTMKL significantly outperforms the state-of-the-art method in [21], yielding a performance in line with [13], due to its ability to share information across tasks.

**Image denoising and dictionary learning.** Here we illustrate denoising on the $256 \times 256$ "house" image used in [19]. Three noise levels (standard deviations 15, 25 and 50) are considered. Following [19], we partition the noisy image in 62,001 overlapping $8 \times 8$ blocks and regard each block as a different task. MTMKL is then run using $M = 64$ "latent blocks", also known as "dictionary elements" (bigger dictionaries do not result in significant performance increase). For the covariance of the latent functions, we consider two possible choices: Either a white covariance function (as in [19]) or an exponential covariance of the form $k_{\mathrm{EXP}}(\mathbf{x}_i, \mathbf{x}_j) = e^{-\frac{|\mathbf{x}_i - \mathbf{x}_j|}{\ell}}$, where $\mathbf{x}$ are the pixel coordinates within each block. The first option is equivalent to placing an independent standard normal prior on each pixel of the dictionary. The second one, on the other hand, introduces correlation between neighboring pixels in the dictionary. Results are shown in Table 4. The exponential covariance clearly enhances performance and produces a more structured dictionary, as can be seen in Figure 3.(a). The Peak-to-Signal Ratio (PSNR) obtained using the proposed approach is comparable to the state-of-the-art results obtained in [19].

**Image inpainting and dictionary learning.** We now address the inpainting problem in color images. Following [19], we consider a color image in which a random 80% of the RGB components are missing. Using an analogous partitioning scheme as in the previous section we obtain 148,836 blocks of size $8 \times 8 \times 3$, each of which is regarded as a different task. A dictionary size of $M = 100$ and a white covariance function (which is used in [19]) are selected. Note that we do not apply any other preprocessing to data or any specific initialization as it is done in [19]. The PSNR of the image

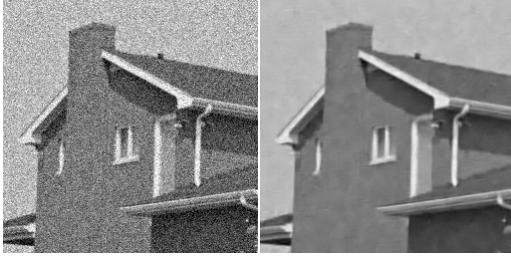

| | PSNR (dB) | | |
|---|---|---|---|
| Noise std | Noisy image | White | Expon. |
| $\sigma = 15$ | 24.66 | 33.98 | 34.29 |
| $\sigma = 25$ | 20.22 | 30.98 | 31.88 |
| $\sigma = 50$ | 14.20 | 26.14 | 28.08 |

Figure 2: Noisy "house" image with $\sigma = 25$ and restored version using Exponential cov. function.

Table 4: PSNR for noisy and restored image using several noise levels and covariance functions.

after it is restored using MTMKL is 28.94 dB, see Figure 3.(b). This result is similar to the results reported in [19] and close to the state-of-the-art result of 29.65 dB achieved in [23].

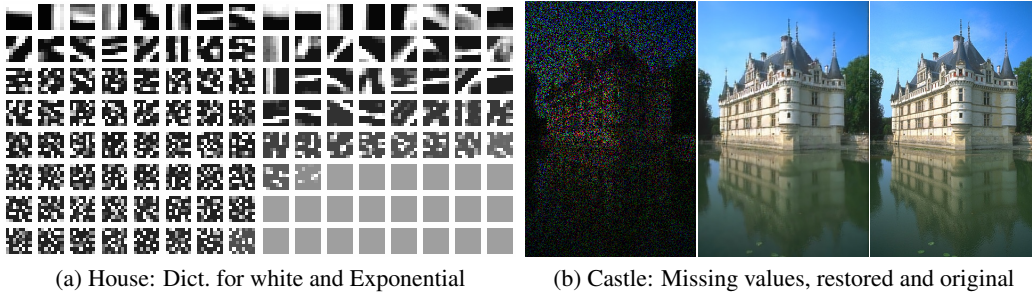

(a) House: Dict. for white and Exponential    (b) Castle: Missing values, restored and original

Figure 3: Dictionaries inferred from noisy ($\sigma = 25$) "house" image; and "castle" inpainting results.

**Collaborative filtering.** Finally, we performed an experiment on the 10M MovieLens data set that consists of 10 million ratings for 71,567 users and 10,681 films, with ratings ranging $\{1, 0.5, 2, \ldots, 4.5, 5\}$. We followed the setup in [24] and used the $r_a$ and $r_b$ partitions provided with the database, that split the data into a training and testing, so that they are 10 ratings per user in the test set. We applied the sparse factor analysis model (i.e. sparse PCA but with heteroscedastic noise for the columns of the observation matrix $\mathbf{Y}$ which corresponds to films) with $M = 20$ latent dimensions. The RMSE for the $r_a$ partition was $0.88$ for the $r_b$ partition was $0.85$ so one average $0.865$. This result is slightly better than $0.8740$ RMSE reported in [24] using GP-LVM.

## 6 Discussion

In this work we have proposed a spike and slab multi-task and multiple kernel learning model. A novel variational algorithm to perform inference in this model has been derived. The key contribution in this regard that explains the good performance of the algorithm is the choice of a joint distribution over $\tilde{w}_{qm}$ and $s_{qm}$ in the variational posterior, as opposed to the usual independence assumption. This has the effect of using exponentially many modes to approximate the posterior, thus rendering it more accurate and much more robust to poor initializations of the variational parameters. The relevance and wide applicability of the proposed model has been illustrated by using it on very diverse tasks: multi-output regression, multi-class classification, image denoising, image inpainting and collaborative filtering. Prior structure beliefs were introduced in image dictionaries, which is also a novel contribution to the best of our knowledge. Finally an interesting topic for future research is to optimize the variational distribution proposed here with alternative approximate inference frameworks such as belief propagation or expectation propagation. This could allow to extend current methodologies within such frameworks that assume unimodal approximations [25, 26].

**Acknowledgments**
We thank the reviewers for insightful comments. MKT was supported by EPSRC Grant No EP/F005687/1 "Gaussian Processes for Systems Identification with Applications in Systems Biology". MLG gratefully acknowledges funding from CAM project CCG10-UC3M/TIC-5511 and CONSOLIDER-INGENIO 2010 CSD2008-00010 (COMONSENS).

# References

[1] T.J. Mitchell and J.J. Beauchamp. Bayesian variable selection in linear regression. *Journal of the American Statistical Association*, 83(404):1023–1032, 1988.

[2] R. Tibshirani. Regression shrinkage and selection via the lasso. *Journal of the Royal Statistical Society, Series B*, 58:267–288, 1994.

[3] M.E. Tipping. Sparse Bayesian learning and the relevance vector machine. *Journal of Machine Learning Research*, 1:211–244, 2001.

[4] E.I. George and R.E. Mcculloch. Variable selection via Gibbs sampling. *Journal of the American Statistical Association*, 88(423):881–889, 1993.

[5] M. West. Bayesian factor regression models in the "large p, small n" paradigm. In *Bayesian Statistics*, pages 723–732. Oxford University Press, 2003.

[6] B. Efron. Microarrays, empirical Bayes and the two-groups model. *Statistical Science*, 23:1–22, 2008.

[7] C. Archambeau and F. Bach. Sparse probabilistic projections. In D. Koller, D. Schuurmans, Y. Bengio, and L. Bottou, editors, *Advances in Neural Information Processing Systems 21*, pages 73–80. 2009.

[8] F. Caron and A. Doucet. Sparse Bayesian nonparametric regression. In *In 25th International Conference on Machine Learning (ICML). ACM*, 2008.

[9] Matthias W. Seeger and Hannes Nickisch. Compressed sensing and Bayesian experimental design. In *ICML*, pages 912–919, 2008.

[10] C.M. Carvalho, N.G. Polson, and J.G. Scott. The horseshoe estimator for sparse signals. *Biometrika*, 97:465–480, 2010.

[11] T. Damoulas and M.A. Girolami. Probabilistic multi-class multi-kernel learning: on protein fold recognition and remote homology detection. *Bioinformatics*, 24:1264–1270, 2008.

[12] M. Christoudias, R. Urtasun, and T. Darrell. Bayesian localized multiple kernel learning. Technical report, EECS Department, University of California, Berkeley, Jul 2009.

[13] C. Archambeau and F. Bach. Multiple Gaussian process models. In *NIPS 23 workshop on New Directions in Multiple Kernel Learning*. 2010.

[14] Y.W. Teh, M. Seeger, and M.I. Jordan. Semiparametric latent factor models. In *Proceedings of the International Workshop on Artificial Intelligence and Statistics*, volume 10, 2005.

[15] E.V. Bonilla, K.M.A. Chai, and C.K.I. Williams. Multi-task Gaussian process prediction. In *Advances Neural Information Processing Systems 20*, 2008.

[16] P Boyle and M. Frean. Dependent Gaussian processes. In *Advances in Neural Information Processing Systems 17*, pages 217–224. MIT Press, 2005.

[17] M. Alvarez and N.D. Lawrence. Sparse convolved Gaussian processes for multi-output regression. In *Advances in Neural Information Processing Systems 20*, pages 57–64, 2008.

[18] R. Yoshida and M. West. Bayesian learning in sparse graphical factor models via variational mean-field annealing. *Journal of Machine Learning Research*, 11:1771–1798, 2010.

[19] M. Zhou, H. Chen, J. Paisley, L. Ren, G. Sapiro, and L. Carin. Non-parametric Bayesian dictionary learning for sparse image represent ations. In Y. Bengio, D. Schuurmans, J. Lafferty, C. K. I. Williams, and A. Culotta, editors, *Advances in Neural Information Processing Systems 22*, pages 2295–2303. 2009.

[20] M. Lázaro-Gredilla and M. Titsias. Variational heteroscedastic Gaussian process regression. In *28th International Conference on Machine Learning (ICML-11)*, pages 841–848, New York, NY, USA, June 2011. ACM.

[21] M.E. Nilsback and A. Zisserman. Automated flower classification over a large number of classes. In *Proceedings of the Indian Conference on Computer Vision, Graphics and Image Processing*, Dec 2008.

[22] M. Varma and D. Ray. Learning the discriminative power invariance trade-off. In *International Conference on Computer Vision*. 2007.

[23] J. Mairal, M. Elad, and G. Sapiro. Sparse representation for color image restoration. *IEEE Trans. Image Processing*, 17, 2008.

[24] N.D. Lawrence and R. Urtasun. Non-linear matrix factorization with Gaussian processes. In *Proceedings of the 26th Annual International Conference on Machine Learning*, pages 601–608, 2009.

[25] K. Sharp and M. Rattray. Dense message passing for sparse principal component analysis. In *13th International Conference on Artificial Intelligence and Statistics (AISTATS)*, pages 725–732, 2010.

[26] J.M. Hernández-Lobato, D. Hernández-Lobato, and A. Suárez. Network-based sparse Bayesian classification. *Pattern Recognition*, 44(4):886–900, 2011.

